# A NEURAL-NETWORK SOLUTION TO THE CONCENTRATOR ASSIGNMENT PROBLEM

Gene A. Tagliarini
Edward W. Page

Department of Computer Science, Clemson University, Clemson, SC
29634-1906

## ABSTRACT

Networks of simple analog processors having neuron-like properties have been employed to compute good solutions to a variety of optimization problems. This paper presents a neural-net solution to a resource allocation problem that arises in providing local access to the backbone of a wide-area communication network. The problem is described in terms of an energy function that can be mapped onto an analog computational network. Simulation results characterizing the performance of the neural computation are also presented.

## INTRODUCTION

This paper presents a neural-network solution to a resource allocation problem that arises in providing access to the backbone of a communication network.[1] In the field of operations research, this problem was first known as the warehouse location problem and heuristics for finding feasible, suboptimal solutions have been developed previously.[2,3] More recently it has been known as the multifacility location problem[4] and as the concentrator assignment problem.[1]

## THE HOPFIELD NEURAL NETWORK MODEL

The general structure of the Hopfield neural network model[5,6,7] is illustrated in Fig. 1. Neurons are modeled as amplifiers that have a sigmoid input/ output curve as shown in Fig. 2. Synapses are modeled by permitting the output of any neuron to be connected to the input of any other neuron. The strength of the synapse is modeled by a resistive connection between the output of a neuron and the input to another. The amplifiers provide integrative analog summation of the currents that result from the connections to other neurons as well as connection to external inputs. To model both excitatory and inhibitory synaptic links, each amplifier provides both a normal output $V$ and an inverted output $\overline{V}$. The normal outputs range between 0 and 1 while the inverting amplifier produces corresponding values between 0 and -1. The synaptic link between the output of one amplifier and the input of another is defined by a conductance $T_{ij}$ which connects one of the outputs of amplifier $j$ to the input of amplifier $i$. In the Hopfield model, the connection between neurons $i$ and $j$ is made with a resistor having a value $R_{ij} = 1/T_{ij}$ . To provide an excitatory synaptic connection (positive $T_{ij}$), the resistor is connected to the normal output of

This research was supported by the U.S. Army Strategic Defense Command.

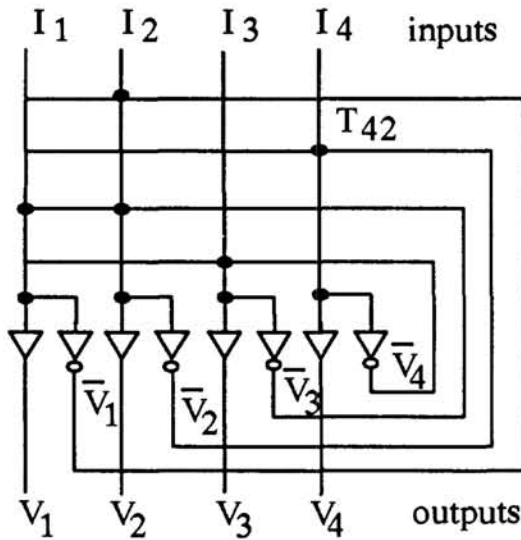

Fig. 1. Schematic for a simplified Hopfield network with four neurons.

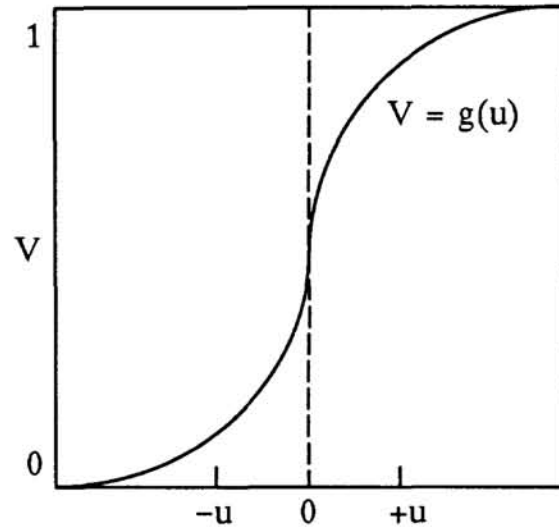

Fig. 2. Amplifier input/output relationship

amplifier j. To provide an inhibitory connection (negative $T_{ij}$), the resistor is connected to the inverted output of amplifier j. The connections among the neurons are defined by a matrix T consisting of the conductances $T_{ij}$. Hopfield has shown that a symmetric T matrix ($T_{ij} = T_{ji}$) whose diagonal entries are all zeros, causes convergence to a stable state in which the output of each amplifier is either 0 or 1. Additionally, when the amplifiers are operated in the high–gain mode, the stable states of a network of n neurons correspond to the local minima of the quantity

$$E = (-1/2) \sum_{i=1}^{n} \sum_{j=1}^{n} T_{ij} V_i V_j \quad - \quad \sum_{i=1}^{n} V_i I_i \qquad (1)$$

where $V_i$ is the output of the $i^{th}$ neuron and $I_i$ is the externally supplied input to the $i^{th}$ neuron. Hopfield refers to E as the computational energy of the system.

## THE CONCENTRATOR ASSIGNMENT PROBLEM

Consider a collection of n sites that are to be connected to m concentrators as illustrated in Fig. 3(a). The sites are indicated by the shaded circles and the concentrators are indicated by squares. The problem is to find an assignment of sites to concentrators that minimizes the total cost of the assignment and does not exceed the capacity of any concentrator. The constraints that must be met can be summarized as follows:

a) Each site i ( i = 1, 2,..., n ) is connected to exactly one concentrator; and

b) Each concentrator j ( j = 1, 2,..., m ) is connected to no more than $k_j$ sites (where $k_j$ is the capacity of concentrator j).

Figure 3(b) illustrates a possible solution to the problem represented in Fig. 3(a).

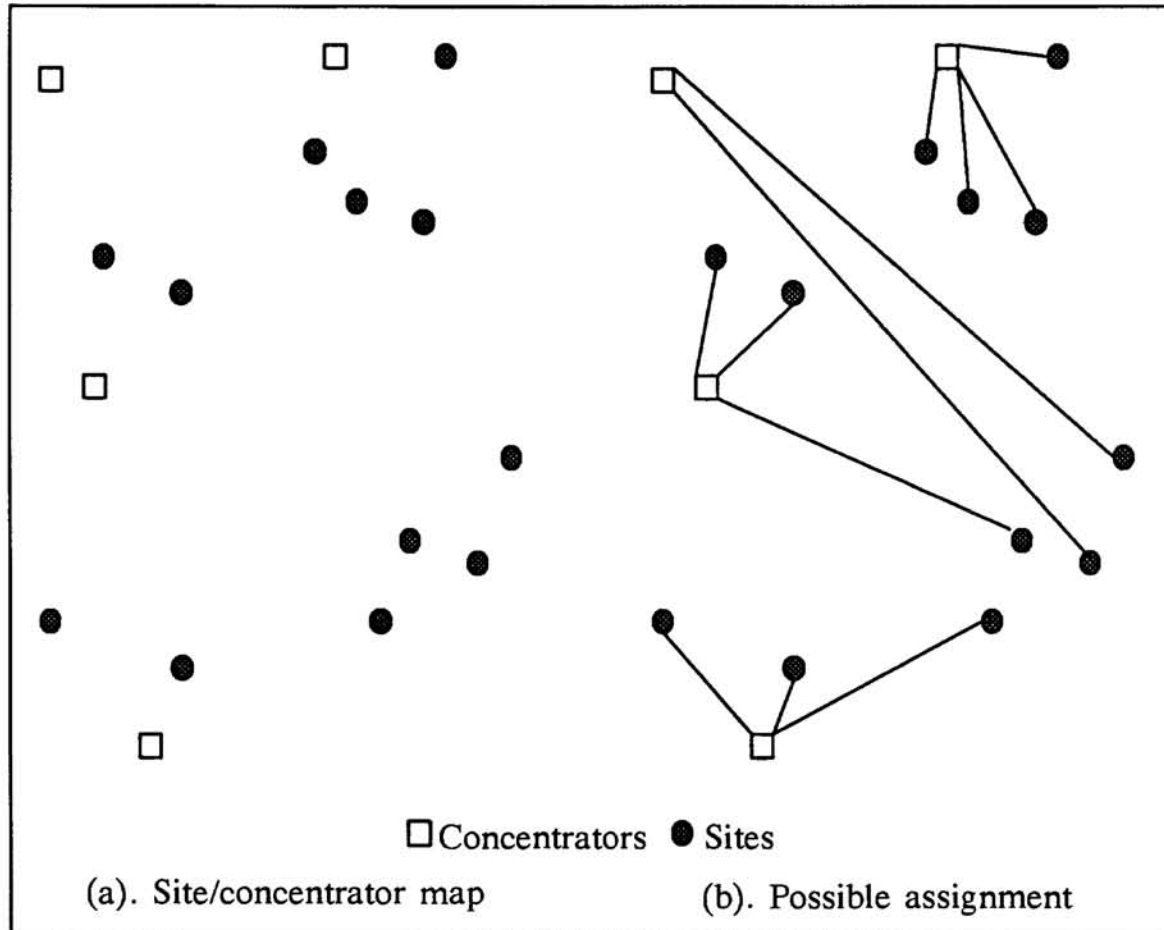

□ Concentrators  ● Sites

(a). Site/concentrator map          (b). Possible assignment

Fig. 3. Example concentrator assignment problem

If the cost of assigning site i to concentrator j is $c_{ij}$ , then the total cost of a particular assignment is

$$\text{total cost} = \sum_{i=1}^{n} \sum_{j=1}^{m} x_{ij} \, c_{ij} \tag{2}$$

where $x_{ij} = 1$ only if we actually decide to assign site i to concentrator j and is 0 otherwise. There are $m^n$ possible assignments of sites to concentrators that satisfy constraint a). Exhaustive search techniques are therefore impractical except for relatively small values of m and n.

## THE NEURAL NETWORK SOLUTION

This problem is amenable to solution using the Hopfield neural network model. The Hopfield model is used to represent a matrix of possible assignments of sites to concentrators as illustrated in Fig. 4. Each square corresponds

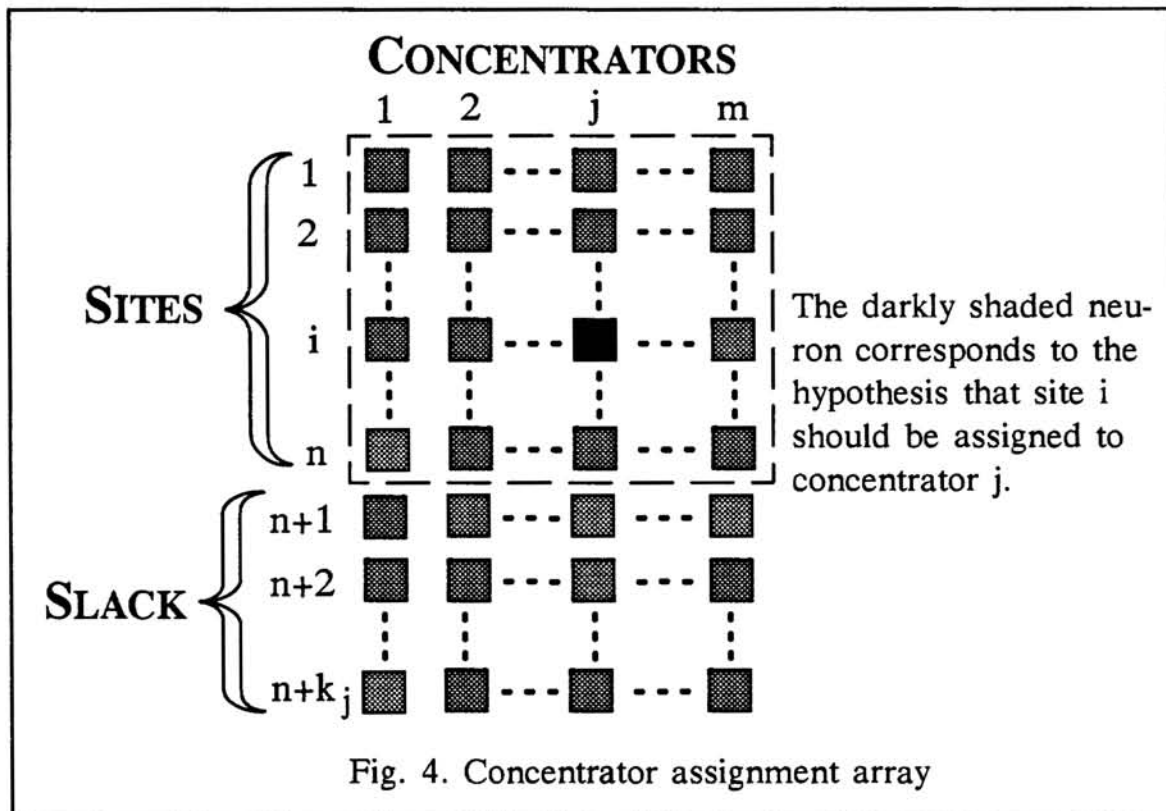

The darkly shaded neuron corresponds to the hypothesis that site i should be assigned to concentrator j.

Fig. 4. Concentrator assignment array

to a neuron and a neuron in row i and column j of the upper n rows of the array represents the hypothesis that site i should be connected to concentrator j. If the neuron in row i and column j is on, then site i should be assigned to concentrator j; if it is off, site i should not be assigned to concentrator j.

The neurons in the lower sub–array, indicated as "SLACK", are used to implement individual concentrator capacity constraints. The number of slack neurons in a column should equal the capacity (expressed as the number sites which can be accommodated) of the corresponding concentrator. While it is not necessary to assume that the concentrators have equal capacities, it was assumed here that they did and that their cumulative capacity is greater than or equal to the number of sites.

To enable the neurons in the network illustrated above to compute solutions to the concentrator problem, the network must realize an energy function in which the lowest energy states correspond to the least cost assignments. The energy function must therefore favor states which satisfy constraints a) and b) above as well as states that correspond to a minimum cost assignment. The energy function is implemented in terms of connection strengths between neurons. The following section details the construction of an appropriate energy function.

## THE ENERGY FUNCTION

Consider the following energy equation:

$$E = A \sum_{i=1}^{n} \left( \sum_{j=1}^{m} V_{ij} - 1 \right)^2 + B \sum_{j=1}^{m} \left( \sum_{i=1}^{n+k_j} V_{ij} - k_j \right)^2 \quad (3)$$

$$+ C \sum_{j=1}^{m} \sum_{i=1}^{n+k_j} V_{ij} (1 - V_{ij})$$

where $V_{ij}$ is the output of the amplifier in row i and column j of the neuron matrix, m and n are the number of concentrators and the number of sites respectively, and $k_j$ is the capacity of concentrator j.

The first term will be minimum when the sum of the outputs in each row of neurons associated with a site equals one. Notice that this term influences only those rows of neurons which correspond to sites; no term is used to coerce the rows of slack neurons into a particular state.

The second term of the equation will be minimum when the sum of the outputs in each column equals the capacity $k_j$ of the corresponding concentrator. The presence of the $k_j$ slack neurons in each column allows this term to enforce the concentrator capacity restrictions. The effect of this term upon the upper sub-array of neurons (those which correspond to site assignments) is that no more than $k_j$ sites will be assigned to concentrator j. The number of neurons to be turned on in column j is $k_j$; consequently, the number of neurons turned on in column j of the assignment sub-array will be less than or equal to $k_j$ .

The third term causes the energy function to favor the "zero" and "one" states of the individual neurons by being minimum when all neurons are in one or the other of these states. This term influences all neurons in the network.

In summary, the first term enforces constraint a) and the second term enforces constraint b) above. The third term guarantees that a choice is actually made; it assures that each neuron in the matrix will assume a final state near zero or one corresponding to the $x_{ij}$ term of the cost equation (Eq. 2).

After some algebraic re-arrangement, Eq. 3 can be written in the form of Eq. 1 where

$$T_{ij,kl} = \begin{cases} A * \delta(i,k) * (1-\delta(j,l)) + B * \delta(j,l) * (1-\delta(i,k)), & \text{if } i{\leq}n \text{ and } k{\leq}n \\ C * \delta(j,l) * (1-\delta(i,k)), & \text{if } i{>}n \text{ or } k{>}n. \end{cases} \quad (4)$$

Here quadruple subscripts are used for the entries in the matrix T. Each entry indicates the strength of the connection between the neuron in row i and column j and the neuron in row k and column l of the neuron matrix. The function delta is given by

$$\delta(i,j) = \begin{cases} 1, \text{ if } i = j \\ 0, \text{ otherwise.} \end{cases} \tag{5}$$

The A and B terms specify inhibitions within a row or a column of the upper sub-array and the C term provides the column inhibitions required for the neurons in the sub-array of slack neurons.

Equation 3 specifies the form of a solution but it does not include a term that will cause the network to favor minimum cost assignments. To complete the formulation, the following term is added to each $T_{ij,kl}$:

$$\frac{D * \delta(j,1) * (1 - \delta(i,k))}{(\text{cost}[i,j] + \text{cost}[k,1])}$$

where cost[ i , j ] is the cost of assigning site i to concentrator j. The effect of this term is to reduce the inhibitions among the neurons that correspond to low cost assignments. The sum of the costs of assigning both site i to concentrator j and site k to concentrator l was used in order to maintain the symmetry of T.

The external input currents were derived from the energy equation (Eq.3) and are given by

$$I_{ij} = \begin{cases} 2 * kj, \text{ if } i \leq n \\ 2 * kj - 1, \text{ otherwise.} \end{cases} \tag{6}$$

This exemplifies a technique for combining external input currents which arise from combinations of certain basic types of constraints.

## AN EXAMPLE

The neural network solution for a concentrator assignment problem consisting of twelve sites and five concentrators was simulated. All sites and concentrators were located within the unit square on a randomly generated map.

For this problem, it was assumed that no more than three sites could be assigned to a concentrator. The assignment cost matrix and a typical assignment resulting from the simulation are shown in Fig. 5. It is interesting to notice that the network proposed an assignment which made no use of concentrator 2.

Because the capacity of each concentrator $k_j$ was assumed to be three sites, the external input current for each neuron in the upper sub-array was

$$I_{ij} = 6$$

while in the sub-array of slack neurons it was

$$I_{ij} = 5.$$

The other parameter values used in the simulation were

$$A = B = C = -2$$

and

$$D = 0.1 .$$

**CONCENTRATORS**

| SITES | 1 | 2 | 3 | 4 | 5 |
|---|---|---|---|---|---|
| A | .47 | .28 | .55 | (.12) | .46 |
| B | .72 | .75 | (.33) | .40 | .63 |
| C | .95 | .71 | (.31) | .39 | .92 |
| D | .88 | .78 | (.06) | .38 | .82 |
| E | .31 | .62 | .81 | .56 | (.21) |
| F | .25 | .51 | .76 | .46 | (.16) |
| G | .17 | .39 | .77 | .41 | (.11) |
| H | (.66) | .81 | .54 | .52 | .56 |
| I | .60 | .67 | .44 | (.36) | .51 |
| J | (.58) | .84 | .76 | .66 | .48 |
| K | .42 | .33 | .55 | (.15) | .38 |
| L | (.19) | .60 | 1.05 | .71 | .18 |

Fig. 5. The concentrator assignment cost matrix with choices circled.

Since this choice of parameters results in a T matrix that is symmetric and whose diagonal entries are all zeros, the network will converge to the minima of Eq. 3. Furthermore, inclusion of the term which is weighted by the parameter D causes the network to favor minimum cost assignments.

To evaluate the performance of the simulated network, an exhaustive search of all solutions to the problem was conducted using a backtracking algorithm. A frequency distribution of the solution costs associated with the assignments generated by the exhaustive search is shown in Fig. 6. For comparison, a histogram of the results of one hundred consecutive runs of the neural–net simulation is shown in Fig. 7. Although the neural–net simulation did not find a global minimum, ninety–two of the one hundred assignments which it did find were among the best 0.01% of all solutions and the remaining eight were among the best 0.3%.

## CONCLUSION

Neural networks can be used to find good, though not necessarily opti-mal, solutions to combinatorial optimization problems like the concentrator

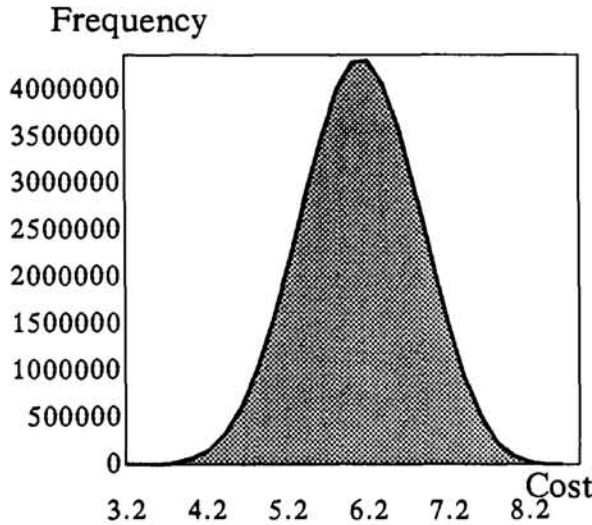

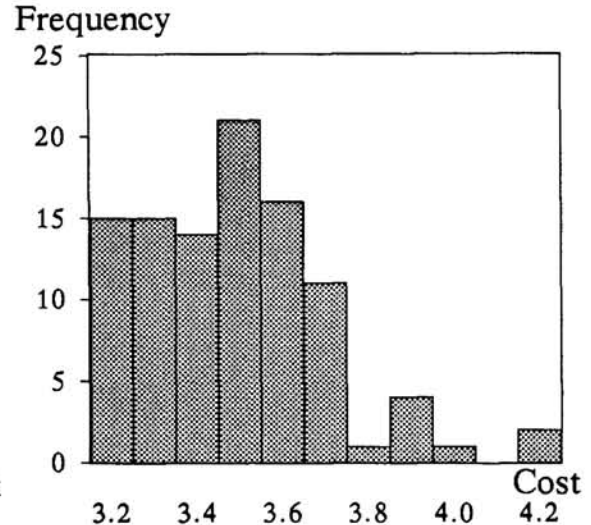

Fig. 6. Distribution of assignment costs resulting from an exhaustive search of all possible solutions.

Fig. 7. Distribution of assignment costs resulting from 100 consecutive executions of the neural net simulation.

assignment problem. In order to use a neural network to solve such problems, it is necessary to be able to represent a solution to the problem as a state of the network. Here the concentrator assignment problem was successfully mapped onto a Hopfield network by associating each neuron with the hypothesis that a given site should be assigned to a particular concentrator. An energy function was constructed to determine the connections that were needed and the resulting neural network was simulated.

While the neural network solution to the concentrator assignment problem did not find a globally minimum cost assignment, it very effectively rejected poor solutions. The network was even able to suggest assignments which would allow concentrators to be removed from the communication network.

## REFERENCES

1. A. S. Tanenbaum, Computer Networks (Prentice–Hall: Englewood Cliffs, New Jersey, 1981), p. 83.

2. E. Feldman, F. A. Lehner and T. L. Ray, Manag. Sci. V12, 670 (1966).

3. A. Kuehn and M. Hamburger, Manag. Sci. V9, 643 (1966).

4. T. Aykin and A. J. G. Babu, J. of the Oper. Res. Soc. V38, N3, 241 (1987).

5. J. J. Hopfield, Proc. Natl. Acad. Sci. U. S. A., V79, 2554 (1982).

6. J. J. Hopfield and D. W. Tank, Bio. Cyber. V52, 141 (1985).

7. D. W. Tank and J. J. Hopfield, IEEE Trans. on Cir. and Sys. CAS–33, N5, 533 (1986).
